# The Entropy Regularization Information Criterion

**Alex J. Smola**
Dept. of Engineering and RSISE
Australian National University
Canberra ACT 0200, Australia
Alex.Smola@anu.edu.au

**John Shawe-Taylor**
Royal Holloway College
University of London
Egham, Surrey TW20 0EX, UK
john@dcs.rhbnc.ac.uk

**Bernhard Schölkopf**
Microsoft Research Limited
St. George House, 1 Guildhall Street
Cambridge CB2 3NH
bsc@microsoft.com

**Robert C. Williamson**
Dept. of Engineering
Australian National University
Canberra ACT 0200, Australia
Bob.Williamson@anu.edu.au

## Abstract

Effective methods of capacity control via uniform convergence bounds for function expansions have been largely limited to Support Vector machines, where good bounds are obtainable by the entropy number approach. We extend these methods to systems with expansions in terms of arbitrary (parametrized) basis functions and a wide range of regularization methods covering the whole range of general linear additive models. This is achieved by a data dependent analysis of the eigenvalues of the corresponding design matrix.

## 1 INTRODUCTION

Model selection criteria based on the Vapnik-Chervonenkis (VC) dimension are known to be difficult to obtain, worst case, and often not very tight. Yet they have the theoretical appeal of providing bounds, with few or no assumptions made.

Recently new methods [8, 7, 6] have been developed which are able to provide a better characterization of the complexity of function classes than the VC dimension, and moreover, are easily obtainable and take advantage of the data at hand (i.e. they employ the concept of luckiness). These techniques, however, have been limited to linear functions or expansions of functions in terms of kernels as happens to be the case in Support Vector (SV) machines.

In this paper we show that the previously mentioned techniques can be extended to expansions in terms of arbitrary basis functions, covering a large range of practical algorithms such as general linear models, weight decay, sparsity regularization [3], and regularization networks [4].

## 2 SUPPORT VECTOR MACHINES

Support Vector machines carry out an effective means of capacity control by minimizing a weighted sum of the training error

$$R_{\mathrm{emp}}[f] := \frac{1}{m} \sum_{i=1}^{m} c(x_i, y_i, f(x_i)) \tag{1}$$

and a regularization term $Q[f] = \frac{1}{2}\|w\|^2$; i.e. they minimize the regularized risk functional

$$R_{\mathrm{reg}}[f] := R_{\mathrm{emp}}[f] + \lambda Q[f] = \frac{1}{m} \sum_{i=1}^{m} c(x_i, y_i, f(x_i)) + \frac{\lambda}{2}\|w\|^2. \tag{2}$$

Here $X := \{x_1, \ldots x_m\} \subset \mathcal{X}$ denotes the training set, $Y := \{y_1, \ldots y_m\} \subset \mathcal{Y}$ the corresponding labels (target values), $\mathcal{X}, \mathcal{Y}$ the corresponding domains, $\lambda > 0$ a regularization constant, $c : \mathcal{X} \times \mathcal{Y} \times \mathcal{Y} \to \mathbb{R}_0^+$ a cost function, and $f : \mathcal{X} \to \mathcal{Y}$ is given by

$$f(x) := \langle x, w \rangle, \text{ or in the nonlinear case } f(x) := \langle \Phi(x), w \rangle. \tag{3}$$

Here $\Phi : \mathcal{X} \to \mathcal{F}$ is a map into a feature space $\mathcal{F}$. Finally, dot products in feature space can be written as $\langle \Phi(x), \Phi(x') \rangle = k(x, x')$ where $k$ is a so–called Mercer kernel.

For $n \in \mathbb{N}$, $\mathbb{R}^n$ denotes the $n$-dimensional space of vectors $x = (x_1, \ldots, x_n)$. We define spaces $\ell_p^n$ as follows: as vector spaces, they are identical to $\mathbb{R}^n$, in addition, they are endowed with $p$-norms:

$$
\begin{aligned}
\|x\|_{\ell_p^n} &:= \|x\|_p = \left( \sum_{j=1}^{n} |x_j|^p \right)^{1/p} && \text{for } 0 < p < \infty \\
\|x\|_{\ell_\infty^n} &:= \max_{j=1,\ldots,n} |x_j| && \text{for } p = \infty
\end{aligned}
$$

We write $\ell_p = \ell_p^\infty$. Furthermore let $U_{\ell_p^n} := \{x : \|x\|_{\ell_p^n} \leq 1\}$ be the unit $\ell_p^n$-ball.

For model selection purposes one wants to obtain bounds on the richness of the map $S_X$

$$S_X : w \mapsto (f(x_1), \ldots, f(x_m)) = (\langle \Phi(x_1), w \rangle, \ldots, \langle \Phi(x_m), w \rangle). \tag{4}$$

where $w$ is restricted to an $\ell_2$ unit ball of some radius $\Lambda$ (this is equivalent to choosing an appropriate value of $\lambda$ — an increase in $\lambda$ decreases $\Lambda$ and vice versa). By the "richness" of $S_X$ specifically we mean the $\ell_\infty^m$ $\epsilon$-covering numbers $\mathcal{N}(\epsilon, S_X(\Lambda U_{\ell_p^m}), \ell_\infty^m)$ of the set $S_X(\Lambda U_{\ell_p^m})$. In the standard COLT notation, we mean

$$\mathcal{N}(\epsilon, S_X(\Lambda U_{\ell_p^m}), \ell_\infty^m) := \min \left\{ n \left| \begin{array}{l} \text{There exists a set } \{z_1, \ldots z_n\} \subset F \text{ such that for all} \\ z \in S_X(\Lambda U_{\ell_p^m}) \text{ we have } \min_{1 \leq i \leq n} \|z - z_i\|_{\ell_\infty^m} \leq \epsilon \end{array} \right. \right\}$$

See [8] for further details.

When carrying out model selection in this case, advanced methods [6] exploit the *distribution* of $X$ mapped into feature space $\mathcal{F}$, and thus of the spectral properties of the operator $S_X$ by analyzing the spectrum of the Gram matrix $G = [g_{ij}]_{ij}$, where $g_{ij} := k(x_i, x_j)$.

All this is possible since $k(x_i, x_j)$ can be seen as a dot product of $x_i, x_j$ mapped into some feature space $\mathcal{F}$, i.e. $k(x_i, x_j) = \langle \Phi(x_i), \Phi(x_j) \rangle$. This property, whilst true for SV machines with Mercer kernels, does not hold in general case where $f$ is expanded in terms of more or less arbitrary basis functions.

## 3  THE BASIC PROBLEMS

One basic problem is that when expanding $f$ into

$$f(x) = \sum_{i=1}^{n} \alpha_i f_i(x) \text{ where } \alpha_i \in \mathbb{R} \tag{5}$$

with $f_i(x)$ being arbitrary functions, it is not immediately obvious how to regard $f$ as a dot product in some feature space. One can show that the VC dimension of a set of $n$ linearly independent functions is $n$. Hence one would intuitively try to restrict the class of admissible models by controlling the number of basis functions $n$ in terms of which $f$ can be expanded.

Now consider an extreme case. In addition to the $n$ basis functions $f_i$ defined previously, we are given $n$ further basis functions $f_i'$, linearly independent of the previous ones, which differ from $f_i$ only on a small domain $\mathcal{X}'$, i.e. $f_i|_{\mathcal{X} \backslash \mathcal{X}'} = f_i'|_{\mathcal{X} \backslash \mathcal{X}'}$. Since this new set of functions is linearly independent, the VC dimension of the joint set is given by $2n$. On the other hand, if hardly any data occurs on the domain $\mathcal{X}'$, one would not notice the difference between $f_i$ and $f_i'$. In other words, the joint system of functions would behave as if we only had the initial system of $n$ basis functions.

An analogous situation occurs if $f_i' = f_i + \epsilon g_i$ where $\epsilon$ is a small constant and $g_i$ was bounded, say, within $[0, 1]$. Again, in this case, the additional effect of the set of functions $f_i'$ would be hardly noticable, but still, the joint set of functions would count as one with VC dimension $2n$. This already indicates, that simply counting the number of basis functions may not be a good idea after all.

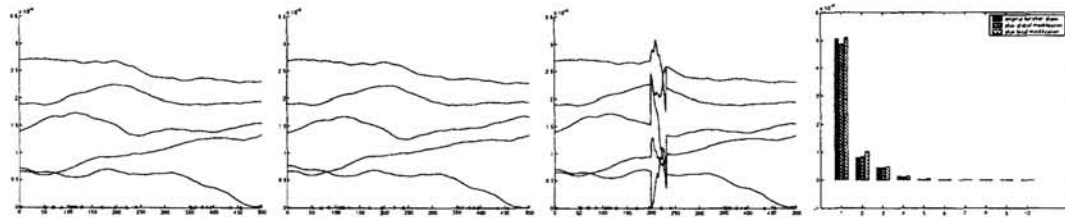

Figure 1: From left to right: (a) initial set of functions $f_1, \ldots, f_5$ (dots on the $x$-axis indicate sampling points); (b) additional set of functions $f_1', \ldots, f_5'$ which differ globally, but only by a small amount; (c) additional set of functions $f_1', \ldots, f_5'$ which differ locally, however by a large amount; (d) spectrum of the corresponding design matrices - the bars denote the cases (a)-(c) in the corresponding order. Note that the difference is quite small.

On the other hand, the spectra of the corresponding design matrices (see Figure 1) are very similar. This suggests the use of the latter for a model selection criterion.

Finally we have the practical problem that capacity control, which in SV machines was carried out by minimizing the length of the "weight vector" $w$ in feature space, cannot be done in an analogous way either. There are several ways to do this. Below we consider three that have appeared in the literature and for which there exist effective algorithms.

**Example 1 (Weight Decay)** *Define $Q[f] := \frac{1}{2} \sum_i \alpha_i^2$; i.e. the coefficients $\alpha_i$ of the function expansion are constrained to an $\ell_2$ ball. In this case we can consider the following operator $S_X^{(1)} : \ell_2^n \to \ell_\infty^m$, where*

$$S_X^{(1)} : \alpha \mapsto (f(x_1), \ldots, f(x_m)) = (\langle f(x_1), \alpha \rangle, \ldots, \langle f(x_m), \alpha \rangle) = F\alpha \tag{6}$$

*Here $f(x) := (f_1(x), \ldots f_n(x))$, $F_{ij} := f_i(x_j)$, $\alpha := (\alpha_1, \ldots, \alpha_n)$ and $\alpha \in \Lambda U_{\ell_2^n}$ for some $\Lambda > 0$.*

**Example 2 (Sparsity Regularization)** *In this case $Q[f] := \sum_i |\alpha_i|$, i.e. the coefficients $\alpha_i$ of the function expansion are constrained to an $\ell_1$ ball to enforce sparseness [3]. Thus $S_X^{(2)} : \ell_1^n \to \ell_\infty^m$ with $S_X^{(2)}$ mapping $\alpha$ as in (6) except $\alpha \in \Lambda U_{\ell_1^n}$. This is similar to expansions encountered in boosting or in linear programming machines.*

**Example 3 (Regularization Networks)** *Finally one could set $Q[f] := \frac{1}{2}\alpha^\top Q \alpha$ for some positive definite matrix $Q$. For instance, $Q_{ij}$ could be obtained from $\langle P f_i, P f_j \rangle$ where $P$ is a regularization operator penalizing non-smooth functions [4]. In this case $\alpha$ lives inside some $n$–dimensional ellipsoid. By substituting $\alpha' := Q^{\frac{1}{2}}\alpha$ one can reduce this setting to the case of example 1 with a different set of basis functions ($f'(x) = Q^{-\frac{1}{2}}f(x)$) and consider an evaluation operator $S_X^{(3)} : \ell_2^n \to \ell_\infty^m$ given by*

$$S_X^{(3)} : \alpha' \mapsto (f(x_1), \ldots, f(x_m)) = (\langle Q^{-\frac{1}{2}}f(x_1), \alpha' \rangle, \ldots, \langle Q^{-\frac{1}{2}}f(x_m), \alpha' \rangle) = Q^{-\frac{1}{2}}F\alpha' \tag{7}$$

*where $\alpha' \in \Lambda U_{\ell_2^n}$ for some $\Lambda > 0$ and $F_{ij} = f_i(x_j)$ as in example 1.*

**Example 4 (Support Vector Machines)** *An important special case of example 3 are Support Vector Machines where we have $Q_{ij} = k(x_i, x_j)$ and $f_i(x) = k(x_i, x)$, hence $Q = F$. Hence the possible values generated by a Support Vector Machine can be written as*

$$S_X^{(4)} : \alpha' \mapsto (f(x_1), \ldots, f(x_m)) = (\langle Q^{-\frac{1}{2}}f(x_1), \alpha' \rangle, \ldots, \langle Q^{-\frac{1}{2}}f(x_m), \alpha' \rangle) = F^{\frac{1}{2}}\alpha' \tag{8}$$

*where $\alpha' \in \Lambda U_{\ell_2^n}$ for some $\Lambda > 0$.*

## 4 ENTROPY NUMBERS

Covering numbers characterize the difficulty of learning elements of a function class. Entropy numbers of operators can be used to compute covering numbers more easily and more tightly than the traditional techniques based on VC-like dimensions such as the fat shattering dimension [1]. Knowing $e_l(S_X) = \epsilon$ (see below for the definition) tells one that $\log \mathcal{N}(\epsilon, F, \ell_\infty^m) \le l$, where $F$ is the effective class of functions used by the regularised learning machines under consideration. In this section we summarize a few basic definitions and results as presented in [8] and [2].

The $l$th entropy number $\epsilon_l(F)$ of a set $F$ with a corresponding metric $d$ is the precision up to which $F$ can be approximated by $l$ elements of $F$; i.e. for all $f \in F$ there exists some $\bar{f}_i \in \{\bar{f}_1, \ldots, \bar{f}_l\}$ such that $d(f, \bar{f}_i) \le \epsilon_l$. Hence $\epsilon_l(F)$ is the functional inverse of the covering number of $F$.

The entropy number of an bounded linear operator $T : A \to B$ between normed linear spaces $A$ and $B$ is defined as $\epsilon_l(T) := \epsilon_l(T(U_A))$ with the metric $d$ being induced by $\|\cdot\|_B$. The *dyadic entropy numbers* $e_l$ are defined by $e_l := \epsilon_{2^{l+1}}$ (the latter quantity is often more convenient to deal with since it corresponds to the log of the covering number).

We make use of the following three results on entropy numbers of the identity mapping from $\ell_{p_1}^n$ into $\ell_{p_2}^n$, diagonal operators, and products of operators. Let

$$\mathrm{id}_{p_1,p_2}^n : \ell_{p_1}^n \to \ell_{p_2}^n \quad ; \quad \mathrm{id}_{p_1,p_2}^n : x \mapsto x$$

The following result is due to Schütt; the constants 9.94 and 1.86 were obtained in [9].

**Proposition 1 (Entropy numbers for identity operators)** *Be $m \in \mathbb{N}$. Then*

$$e_l(\mathrm{id}_{1,2}^n) \le 9.94 \left(\frac{1}{l}\log\left(1 + \frac{n}{l}\right)\right)^{\frac{1}{2}} \quad \& \quad e_l(\mathrm{id}_{2,\infty}^n) \le 1.86 \left(\frac{1}{l}\log\left(1 + \frac{n}{l}\right)\right)^{\frac{1}{2}} \tag{9}$$

**Proposition 2 (Carl and Stephani [2, p. 11])** *Let $E, F, G$ be Banach spaces, $R : F \rightarrow G$, and $S : E \rightarrow F$. Then, for $n, t \in \mathbb{N}$,*

$$e_{n+t-1}(RS) \leq e_n(R)e_t(S), \quad e_n(RS) \leq e_n(R)\|S\| \text{ and } e_n(RS) \leq e_n(S)\|R\|. \quad (10)$$

*Note that the latter two inequalities follow directly from the fact that $\epsilon_1(R) = \|R\|$ for all $R : F \rightarrow G$ by definition of the operator norm $\|R\|$.*

**Proposition 3** *Let $\sigma_1 \geq \sigma_2 \geq \cdots \geq \sigma_j \geq \cdots \geq 0$, $1 \leq p \leq \infty$ and*

$$Dx = (\sigma_1 x_1, \sigma_2 x_2, \ldots, \sigma_j x_j, \ldots) \quad (11)$$

*for $x = (x_1, x_2, \ldots, x_j, \ldots) \in \ell_p$ be the diagonal operator from $\ell_p$ into itself, generated by the sequence $(\sigma_j)_j$. Then for all $n \in \mathbb{N}$,*

$$\sup_{j \in \mathbb{N}} 2^{-\frac{n-1}{j}}(\sigma_1 \sigma_2 \cdots \sigma_j)^{\frac{1}{j}} \leq e_n(D) \leq 6 \sup_{j \in \mathbb{N}} 2^{-\frac{n-1}{j}}(\sigma_1 \sigma_2 \cdots \sigma_j)^{\frac{1}{j}}. \quad (12)$$

## 5  THE MAIN RESULT

We can now state the main theorem which gives bounds on the entropy numbers of $S_X^{(i)}$ for the first three examples of model selection described above (since Support Vector Machines are a special case of example 3 we will not deal with it separately).

**Proposition 4** *Let $f$ be expanded in a linear combination of basis functions as $f := \sum_{i=1}^n \alpha_i f_i$ and the coefficients $\alpha$ restricted to one of the convex sets as described in the examples 1 to 3. Moreover denote by $F_{ij} := f_j(x_i)$ the design matrix on a particular sample $X$, and by $Q$ the regularization matrix in the case of example 3. Then the following bound on $S_X$ holds.*

1. *In the case of weight decay (ex. 1) (with $l_1 + l_2 \geq l + 1$)*

$$e_l(S_X^{(1)}) \leq 1.96 \left(l_1^{-1} \log\left(1 + m/l_1\right)\right)^{\frac{1}{2}} e_{l_2}(\Sigma). \quad (13)$$

2. *In the case of weight sparsity regularization (ex. 2) (with $l_1 + l_2 + l_3 \geq l + 2$)*

$$e_l(S_X^{(2)}) \leq 18.48 \left(l_1^{-1} \log\left(1 + m/l_1\right)\right)^{\frac{1}{2}} e_{l_2}(\Sigma) \left(l_3^{-1} \log\left(1 + m/l_3\right)\right)^{\frac{1}{2}}. \quad (14)$$

3. *Finally, in the case of regularization networks (ex. 3) (with $l_1 + l_2 \geq l + 1$)*

$$e_l(S_X^{(3)}) \leq 1.96 \left(l_1^{-1} \log\left(1 + m/l_1\right)\right)^{\frac{1}{2}} e_{l_2}(\Sigma). \quad (15)$$

*Here $\Sigma$ is a diagonal scaling operator (matrix) with $(i, i)$ entries $\sqrt{\sigma_i}$ and $(\sqrt{\sigma_i})_i$ are the eigenvalues (sorted in decreasing order) of the matrix $FF^\top$ in the case of examples 1 and 2, and $FQ^{-1}F^\top$ in the case of example 3.*

The entropy number of $\Sigma$ is readily bounded in terms of $(\sigma_i)_i$ by using (3). One can see that the first setting (weight decay) is a special case of the third one, namely when $Q = \mathbf{1}$, i.e. when $Q$ is just the identity matrix.

**Proof** The proof relies on a factorization of $S_X^{(i)}$ ($i = 1, 2, 3$) in the following way. First we consider the equivalent operator $\hat{S}_X$ mapping from $\ell_2^n$ to $\ell_2^m$ and perform a singular value decomposition [5] of the latter into $\hat{S}_X = V\Sigma W$ where $V, W$ are operators of norm 1, and $\Sigma$ contains the singular values of $S_X^{(i)}$, i.e. the singular values of $F$ and $FQ^{-\frac{1}{2}}$

respectively. The latter, however, are identical to the square root of the eigenvalues of $FF^\top$ or $FQ^{-1}F^\top$. Consequently we can factorize $S_X^{(i)}$ as in the diagram

$$
\begin{array}{ccccc}
& & S_X^{(2)} & & \\
& \overset{id}{\longrightarrow} & \ell_2^n & \overset{\{S_X^{(1)}, S_X^{(3)}\}}{\longrightarrow} & \ell_\infty^m \\
\ell_1^n & & \downarrow W & & \uparrow id \\
& & \ell_2^m & \overset{\Sigma}{\longrightarrow} \ell_2^m \overset{V}{\longrightarrow} & \ell_2^m
\end{array}
\qquad (16)
$$

Finally, in order to compute the entropy number of the overall operator one only has to use the factorization of $S_X$ into $S_X^{(i)} = \mathrm{id}_{2,\infty}^m V\Sigma W$ for $i \in \{1,3\}$ and into $S_X^{(2)} = \mathrm{id}_{2,\infty}^m V\Sigma W \mathrm{id}_{1,2}^n$ for example 2, and apply Proposition 2 several times. We also exploit the fact that for singular value decompositions $\|V\|, \|W\| \leq 1$. ∎

The present theorem allows us to compute the entropy numbers (and thus the complexity) of a class of functions on the current sample $X$. Going back to the examples of section 3, which led to large bounds on the VC dimension one can see that the new result is much less susceptible to such modifications: the addition of $f_1', \ldots f_n'$ to $f_1, \ldots f_n$ does not change the eigenspectrum $\Sigma$ of the design matrix significantly (possibly only doubling the nominal value of the singular values), if the functions $f_i'$ differ from $f_i$ only slightly. Consequently also the bounds will not change significantly even though the number of basis functions just doubled.

Also note that the current error bounds reduce to the results of [6] in the SV case: here $Q_{ij} = F_{ij} = k(x_i, x_j)$ (both the design matrix $F$ and the regularization matrix $Q$ are determined by kernels) and therefore $FQ^{-1}F = Q$. Thus the analysis of the singular values of $FQ^{-1}F$ leads to an analysis of the eigenvalues of the kernel matrix, which is exactly what is done when dealing with SV machines.

## 6 ERROR BOUNDS

To use the above result we need a bound on the expected error of a hypothesis $f$ in terms of the empirical error (training error) and the observed entropy numbers $\epsilon_n(\mathcal{F})$. We use [6, Theorem 4.1] with a small modification.

**Theorem 1** *Let $\mathcal{F}$ be a set of linear functions as described in the previous examples with $e_n(S_X)$ as the corresponding bound on the observed entropy numbers of $\mathcal{F}$ on the dataset $X$. Moreover suppose that for a fixed threshold $b \in \mathbb{R}$ for some $f \in \mathcal{F}$, $\mathrm{sgn}(f - b)$ correctly classifies the set $X$ with a margin $\gamma := \min_{1 \leq i \leq m} |f(x_i) - b|$.*

*Finally let $U := \min\{n \in \mathbb{N} \text{ with } e_n(S_X) \leq \gamma/8.001\}$ and $\alpha(U, \delta) := 3.08(1 + \frac{1}{U}\ln\frac{1}{\delta})$. Then with confidence $1 - \delta$ over $X$ (drawn randomly from $P^m$ where $P$ is some probability distribution) the expected error of $\mathrm{sgn}(f - b)$ is bounded from above by*

$$
\epsilon(m, U, \delta) = \frac{2}{m}\left(U\left(1 + \alpha\left(U, \frac{\delta}{2}\right)\log\left(\frac{5em}{U}\right)\log(17m)\right) + \log\left(\frac{16m}{\delta}\right)\right). \qquad (17)
$$

The proof is essentially identical to that of [6, Theorem 4.1] and is omitted. [6] also shows how to compute $e_n(S_X)$ efficiently including an explicit formula for evaluating $e_l(\Sigma)$.

## 7 DISCUSSION

We showed how improved bounds could be obtained on the entropy numbers of a wide class of popular statistical estimators ranging from weight decay to sparsity regularization

(with SV machines being a special case thereof). The results are given in a way that is directly useable for practicioners without any tedious calculations of the VC dimension or similar combinatorial quantities. In particular, our method ignores (nearly) linear dependent basis functions automatically. Finally, it takes advantage of favourable distributions of data by using the *observed* entropy numbers as a base for stating bounds on the true entropy numbers with respect to the function class under consideration.

Whilst this leads to significantly improved bounds (we achieved an improvement of approximately two orders of magnitude over previous VC-type bounds involving only the radius of the data $R$ and the weight vector $\|w\|$ in the experiments) on the expected risk, the bounds are still not good enough to become predictive. This indicates that possibly rather than using the standard uniform convergence bounds (as used in the previous section) one might want to use other techniques such as a PAC-Bayesian treatment (as recently suggested by Herbrich and Graepel) in combination with the bounds on eigenvalues of the design matrix.

**Acknowledgements:** This work was supported by the Australian Research Council and a grant of the Deutsche Forschungsgemeinschaft SM 62/1-1.

# References

[1] N. Alon, S. Ben-David, N. Cesa-Bianchi, and D. Haussler. Scale–sensitive Dimensions, Uniform Convergence, and Learnability. *J. of the ACM*, 44(4):615–631, 1997.

[2] B. Carl and I. Stephani. *Entropy, compactness, and the approximation of operators.* Cambridge University Press, Cambridge, UK, 1990.

[3] S. Chen, D. Donoho, and M. Saunders. Atomic decomposition by basis pursuit. Technical Report 479, Department of Statistics, Stanford University, 1995.

[4] F. Girosi, M. Jones, and T. Poggio. Regularization theory and neural networks architectures. *Neural Computation*, 7:219–269, 1995.

[5] R. A. Horn and C. R. Johnson. *Matrix Analysis.* Cambridge University Press, Cambridge, 1992.

[6] B. Schölkopf, J. Shawe-Taylor, A. J. Smola, and R. C. Williamson. Generalization bounds via eigenvalues of the gram matrix. Technical Report NC-TR-99-035, NeuroColt2, University of London, UK, 1999.

[7] J. Shawe-Taylor and R. C. Williamson. Generalization performance of classifiers in terms of observed covering numbers. In *Proc. EUROCOLT'99*, 1999.

[8] R. C. Williamson, A. J. Smola, and B. Schölkopf. Generalization performance of regularization networks and support vector machines via entropy numbers of compact operators. NeuroCOLT NC-TR-98-019, Royal Holloway College, 1998.

[9] R. C. Williamson, A. J. Smola, and B. Schölkopf. A Maximum Margin Miscellany. Typescript, 1999.